# Hamming Distance Metric Learning

**Mohammad Norouzi**[†]      **David J. Fleet**[†]      **Ruslan Salakhutdinov**[†,‡]

Departments of Computer Science[†] and Statistics[‡]
University of Toronto
[norouzi,fleet,rsalakhu]@cs.toronto.edu

## Abstract

Motivated by large-scale multimedia applications we propose to learn mappings from high-dimensional data to binary codes that preserve semantic similarity. Binary codes are well suited to large-scale applications as they are storage efficient and permit exact sub-linear $k$NN search. The framework is applicable to broad families of mappings, and uses a flexible form of triplet ranking loss. We overcome discontinuous optimization of the discrete mappings by minimizing a piecewise-smooth upper bound on empirical loss, inspired by latent structural SVMs. We develop a new loss-augmented inference algorithm that is quadratic in the code length. We show strong retrieval performance on CIFAR-10 and MNIST, with promising classification results using no more than $k$NN on the binary codes.

## 1 Introduction

Many machine learning algorithms presuppose the existence of a pairwise similarity measure on the input space. Examples include semi-supervised clustering, nearest neighbor classification, and kernel-based methods, When similarity measures are not given *a priori*, one could adopt a generic function such as Euclidean distance, but this often produces unsatisfactory results. The goal of metric learning techniques is to improve matters by incorporating side information, and optimizing parametric distance functions such as the Mahalanobis distance [7, 12, 30, 34, 36].

Motivated by large-scale multimedia applications, this paper advocates the use of discrete mappings, from input features to binary codes. Compact binary codes are remarkably storage efficient, allowing one to store massive datasets in memory. The Hamming distance, a natural similarity measure on binary codes, can be computed with just a few machine instructions per comparison. Further, it has been shown that one can perform exact nearest neighbor search in Hamming space significantly faster than linear search, with sublinear run-times [15, 25]. By contrast, retrieval based on Mahalanobis distance requires approximate nearest neighbor (ANN) search, for which state-of-the-art methods (*e.g.*, see [18, 23]) do not always perform well, especially with massive, high-dimensional datasets when storage overheads and distance computations become prohibitive.

Most approaches to discrete (binary) embeddings have focused on preserving the metric (*e.g.* Euclidean) structure of the input data, the canonical example being locality-sensitive hashing (LSH) [4, 17]. Based on random projections, LSH and its variants (*e.g.*, [26]) provide guarantees that metric similarity is preserved for sufficiently long codes. To find *compact* codes, recent research has turned to machine learning techniques that optimize mappings for specific datasets (*e.g.*, [20, 28, 29, 32, 3]). However, most such methods aim to preserve Euclidean structure (*e.g.* [13, 20, 35]).

In metric learning, by comparison, the goal is to preserve semantic structure based on labeled attributes or parameters associated with training exemplars. There are papers on learning binary hash functions that preserve semantic similarity [29, 28, 32, 24], but most have only considered ad hoc datasets and uninformative performance measures, for which it is difficult to judge performance with anything but the qualitative appearance of retrieval results. The question of whether or not it is possible to learn hash functions capable of preserving complex semantic structure, with high fidelity, has remained unanswered.

To address this issue, we introduce a framework for learning a broad class of binary hash functions based on a triplet ranking loss designed to preserve relative similarity (*c.f.* [11, 5]). While certainly useful for preserving metric structure, this loss function is very well suited to the preservation of semantic similarity. Notably, it can be viewed as a form of local ranking loss. It is more flexible than the pairwise hinge loss of [24], and is shown below to produce superior hash functions.

Our formulation is inspired by latent SVM [10] and latent structural SVM [37] models, and it generalizes the minimal loss hashing (MLH) algorithm of [24]. Accordingly, to optimize hash function parameters we formulate a continuous upper-bound on empirical loss, with a new form of loss-augmented inference designed for efficient optimization with the proposed triplet loss on the Hamming space. To our knowledge, this is one of the most general frameworks for learning a broad class of hash functions. In particular, many previous loss-based techniques [20, 24] are not capable of optimizing mappings that involve non-linear projections, *e.g.*, by neural nets.

Our experiments indicate that the framework is capable of preserving semantic structure on challenging datasets, namely, MNIST [1] and CIFAR-10 [19]. We show that $k$-nearest neighbor ($k$NN) search on the resulting binary codes retrieves items that bear remarkable similarity to a given query item. To show that the binary representation is rich enough to capture salient semantic structure, as is common in metric learning, we also report classification performance on the binary codes. Surprisingly, on these datasets, simple $k$NN classifiers in Hamming space are competitive with sophisticated discriminative classifiers, including SVMs and neural networks. An important appeal of our approach is the scalability of $k$NN search on binary codes to billions of data points, and of $k$NN classification to millions of class labels.

## 2   Formulation

The task is to learn a mapping $b(\mathbf{x})$ that projects $p$-dimensional real-valued inputs $\mathbf{x} \in \mathbb{R}^p$ onto $q$-dimensional binary codes $\mathbf{h} \in \mathcal{H} \equiv \{-1, 1\}^q$, while preserving some notion of similarity. This mapping, referred to as a hash function, is parameterized by a real-valued vector $\mathbf{w}$ as

$$b(\mathbf{x}; \mathbf{w}) = \text{sign}\left(f(\mathbf{x}; \mathbf{w})\right), \tag{1}$$

where $\text{sign}(.)$ denotes the element-wise sign function, and $f(\mathbf{x}; \mathbf{w}) : \mathbb{R}^p \to \mathbb{R}^q$ is a real-valued transformation. Different forms of $f$ give rise to different families of hash functions:

1. A linear transform $f(\mathbf{x}) = W\mathbf{x}$, where $W \in \mathbb{R}^{q \times p}$ and $\mathbf{w} \equiv \text{vec}(W)$, is the simplest and most well-studied case [4, 13, 24, 33]. Under this mapping the $k^{th}$ bit is determined by a hyperplane in the input space whose normal is given by the $k^{th}$ row of $W$. [1]

2. In [35], linear projections are followed by an element-wise cosine transform, *i.e.* $f(\mathbf{x}) = \cos(W\mathbf{x})$. For such mappings the bits correspond to stripes of $1$ and $-1$ regions, oriented parallel to the corresponding hyperplanes, in the input space.

3. Kernelized hash functions [20, 21].

4. More complex hash functions are obtained with multilayer neural networks [28, 32]. For example, a two-layer network with a $p'$-dimensional hidden layer and weight matrices $W_1 \in \mathbb{R}^{p' \times p}$ and $W_2 \in \mathbb{R}^{q \times p'}$ can be expressed as $f(\mathbf{x}) = \tanh(W_2 \ \tanh(W_1\mathbf{x}))$, where $\tanh(.)$ is the element-wise hyperbolic tangent function.

Our Hamming distance metric learning framework applies to all of the above families of hash functions. The only restriction is that $f$ must be differentiable with respect to its parameters, so that one is able to compute the Jacobian of $f(\mathbf{x}; \mathbf{w})$ with respect to $\mathbf{w}$.

### 2.1   Loss functions

The choice of loss function is crucial for learning good similarity measures. To this end, most existing supervised binary hashing techniques [13, 22, 24] formulate learning objectives in terms of pairwise similarity, where pairs of inputs are labelled as either similar or dissimilar. Similarity-preserving hashing aims to ensure that Hamming distances between binary codes for similar (dissimilar) items are small (large). For example, MLH [24] uses a *pairwise hinge loss* function. For

two binary codes $\mathbf{h}, \mathbf{g} \in \mathcal{H}$ with Hamming distance[2] $\|\mathbf{h}-\mathbf{g}\|_H$, and a similarity label $s \in \{0, 1\}$, the pairwise hinge loss is defined as:

$$\ell_{\text{pair}}(\mathbf{h}, \mathbf{g}, s) \;=\; \begin{cases} \left[\, \|\mathbf{h}-\mathbf{g}\|_H - \rho + 1 \,\right]_+ & \text{for } s = 1 \quad \textit{(similar)} \\ \left[\, \rho - \|\mathbf{h}-\mathbf{g}\|_H + 1 \,\right]_+ & \text{for } s = 0 \quad \textit{(dissimilar)} \,, \end{cases} \qquad (2)$$

where $[\alpha]_+ \equiv \max(\alpha, 0)$, and $\rho$ is a Hamming distance threshold that separates similar from dissimilar codes. This loss incurs zero cost when a pair of similar inputs map to codes that differ by less than $\rho$ bits. The loss is zero for dissimilar items whose Hamming distance is more than $\rho$ bits.

One problem with such loss functions is that finding a suitable threshold $\rho$ with cross-validation is slow. Furthermore, for many problems one cares more about the relative magnitudes of pairwise distances than their precise numerical values. So, constraining pairwise Hamming distances over all pairs of codes with a single threshold is overly restrictive. More importantly, not all datasets are amenable to labeling input pairs as similar or dissimilar. One way to avoid some of these problems is to define loss in terms of *relative similarity*. Such loss functions have been used in metric learning [5, 11], and, as shown below, they are also naturally suited to Hamming distance metric learning.

To define relative similarity, we assume that the training data includes triplets of items $(\mathbf{x}, \mathbf{x}^+, \mathbf{x}^-)$, such that the pair $(\mathbf{x}, \mathbf{x}^+)$ is more similar than the pair $(\mathbf{x}, \mathbf{x}^-)$. Our goal is to learn a hash function $b$ such that $b(\mathbf{x})$ is closer to $b(\mathbf{x}^+)$ than to $b(\mathbf{x}^-)$ in Hamming distance. Accordingly, we propose a *ranking loss* on the triplet of binary codes $(\mathbf{h}, \mathbf{h}^+, \mathbf{h}^-)$, obtained from $b$ applied to $(\mathbf{x}, \mathbf{x}^+, \mathbf{x}^-)$:

$$\ell_{\text{triplet}}(\mathbf{h}, \mathbf{h}^+, \mathbf{h}^-) \;=\; \left[\, \|\mathbf{h}-\mathbf{h}^+\|_H \,-\, \|\mathbf{h}-\mathbf{h}^-\|_H \,+\, 1 \,\right]_+ \,. \qquad (3)$$

This loss is zero when the Hamming distance between the more-similar pair, $\|\mathbf{h}-\mathbf{h}^+\|_H$, is at least one bit smaller than the Hamming distance between the less-similar pair, $\|\mathbf{h}-\mathbf{h}^-\|_H$. This loss function is more flexible than the pairwise loss function $\ell_{\text{pair}}$, as it can be used to preserve rankings among similar items, for example based on Euclidean distance, or perhaps using path distance between category labels within a phylogenetic tree.

## 3 Optimization

Given a training set of triplets, $\mathcal{D} = \left\{ (\mathbf{x}_i, \mathbf{x}_i^+, \mathbf{x}_i^-) \right\}_{i=1}^n$, our objective is the sum of the empirical loss and a simple regularizer on the vector of unknown parameters $\mathbf{w}$:

$$\mathcal{L}(\mathbf{w}) \;=\; \sum_{(\mathbf{x}, \mathbf{x}^+, \mathbf{x}^-) \in \mathcal{D}} \ell_{\text{triplet}}\big( b(\mathbf{x}; \mathbf{w}), \, b(\mathbf{x}^+; \mathbf{w}), \, b(\mathbf{x}^-; \mathbf{w}) \big) \;+\; \frac{\lambda}{2} \|\mathbf{w}\|_2^2 \,. \qquad (4)$$

This objective is discontinuous and non-convex. The hash function is a discrete mapping and empirical loss is piecewise constant. Hence optimization is very challenging. We cannot overcome the non-convexity, but the problems owing to the discontinuity can be mitigated through the construction of a continuous upper bound on the loss.

The upper bound on loss that we adopt is inspired by previous work on latent structural SVMs [37]. The key observation that relates our Hamming distance metric learning framework to structured prediction is as follows,

$$\begin{aligned} b(\mathbf{x}; \mathbf{w}) \;&=\; \operatorname{sign}\left( f(\mathbf{x}; \mathbf{w}) \right) \\ &=\; \operatorname*{argmax}_{\mathbf{h} \in \mathcal{H}} \, \mathbf{h}^\mathsf{T} f(\mathbf{x}; \mathbf{w}) \,, \end{aligned} \qquad (5)$$

where $\mathcal{H} \equiv \{-1, +1\}^q$. The argmax on the RHS effectively means that for dimensions of $f(\mathbf{x}; \mathbf{w})$ with positive values, the optimal code should take on values $+1$, and when elements of $f(\mathbf{x}; \mathbf{w})$ are negative the corresponding bits of the code should be $-1$. This is identical to the $\operatorname{sign}$ function.

### 3.1 Upper bound on empirical loss

The upper bound on loss that we exploit for learning hash functions takes the following form:

$$\begin{aligned} \ell_{\text{triplet}}\big( b(\mathbf{x}; \mathbf{w}), \, &b(\mathbf{x}^+; \mathbf{w}), \, b(\mathbf{x}^-; \mathbf{w}) \big) \;\leq\; \\ &\max_{\mathbf{g}, \mathbf{g}^+, \mathbf{g}^-} \left\{ \ell_{\text{triplet}}\big( \mathbf{g}, \, \mathbf{g}^+, \, \mathbf{g}^- \big) + \mathbf{g}^\mathsf{T} f(\mathbf{x}; \mathbf{w}) + {\mathbf{g}^+}^\mathsf{T} f(\mathbf{x}^+; \mathbf{w}) + {\mathbf{g}^-}^\mathsf{T} f(\mathbf{x}^-; \mathbf{w}) \right\} \\ &- \max_{\mathbf{h}} \left\{ \mathbf{h}^\mathsf{T} f(\mathbf{x}; \mathbf{w}) \right\} - \max_{\mathbf{h}^+} \left\{ {\mathbf{h}^+}^\mathsf{T} f(\mathbf{x}^+; \mathbf{w}) \right\} - \max_{\mathbf{h}^-} \left\{ {\mathbf{h}^-}^\mathsf{T} f(\mathbf{x}^-; \mathbf{w}) \right\} \,, \quad (6) \end{aligned}$$

where $\mathbf{g}$, $\mathbf{g}^+$, $\mathbf{g}^-$, $\mathbf{h}$, $\mathbf{h}^+$, and $\mathbf{h}^-$ are constrained to be $q$-dimensional binary vectors. To prove the inequality in Eq. 6, note that if the first term on the RHS were maximized[3] by $(\mathbf{g}, \mathbf{g}^+, \mathbf{g}^-) = (b(\mathbf{x}), b(\mathbf{x}^+), b(\mathbf{x}^-))$, then using Eq. 5, it is straightforward to show that Eq. 6 would become an equality. In all other cases of $(\mathbf{g}, \mathbf{g}^+, \mathbf{g}^-)$ which maximize the first term, the RHS can only be as large or larger than when $(\mathbf{g}, \mathbf{g}^+, \mathbf{g}^-) = (b(\mathbf{x}), b(\mathbf{x}^+), b(\mathbf{x}^-))$, hence the inequality holds.

Summing the upper bound instead of the loss in Eq. 4 yields an upper bound on the regularized empirical loss in Eq. 4. Importantly, the resulting bound is easily shown to be continuous and piecewise smooth in $\mathbf{w}$ as long as $f$ is continuous in $\mathbf{w}$. The upper bound of Eq. 6 is a generalization of a bound introduced in [24] for the linear case, $f(\mathbf{x}) = W\mathbf{x}$. In particular, when $f$ is linear in $\mathbf{w}$, the bound on regularized empirical loss becomes piecewise linear and convex-concave. While the bound in Eq. 6 is more challenging to optimize than the bound in [24], it allows us to learn hash functions based on non-linear functions $f$, *e.g.* neural networks. While the bound in [24] was defined for $\ell_{\text{pair}}$-type loss functions and pairwise similarity labels, the bound here applies to the more flexible class of triplet loss functions.

## 3.2 Loss-augmented inference

To use the upper bound in Eq. 6 for optimization, we must be able to find the binary codes given by

$$(\hat{\mathbf{g}}, \hat{\mathbf{g}}^+, \hat{\mathbf{g}}^-) \;=\; \underset{(\mathbf{g},\mathbf{g}^+,\mathbf{g}^-)}{\operatorname{argmax}} \left\{ \ell_{\text{triplet}}\big(\mathbf{g}, \mathbf{g}^+, \mathbf{g}^-\big) + \mathbf{g}^\mathsf{T} f(\mathbf{x}) + {\mathbf{g}^+}^\mathsf{T} f(\mathbf{x}^+) + {\mathbf{g}^-}^\mathsf{T} f(\mathbf{x}^-) \right\}. \quad (7)$$

In the structured prediction literature this maximization is called loss-augmented inference. The challenge stems from the $2^{3q}$ possible binary codes over which one has to maximize the RHS.

Fortunately, we can show that this loss-augmented inference problem can be solved efficiently for the class of triplet loss functions that depend only on the value of

$$d(\mathbf{g}, \mathbf{g}^+, \mathbf{g}^-) \;\equiv\; \|\mathbf{g}-\mathbf{g}^+\|_H \,-\, \|\mathbf{g}-\mathbf{g}^-\|_H \,.$$

Importantly, such loss functions do not depend on the specific binary codes, but rather just the differences. Further, note that $d(\mathbf{g}, \mathbf{g}^+, \mathbf{g}^-)$ can take on only $2q+1$ possible values, since it is an integer between $-q$ and $+q$. Clearly the triplet ranking loss only depends on $d$ since

$$\ell_{\text{triplet}}\big(\mathbf{g}, \mathbf{g}^+, \mathbf{g}^-\big) \;=\; \ell'\big(d(\mathbf{g}, \mathbf{g}^+, \mathbf{g}^-)\big), \quad \text{where} \quad \ell'(\alpha) = [\,\alpha - 1\,]_+ \,. \quad (8)$$

For this family of loss functions, given the values of $f(.)$ in Eq. 7, loss-augmented inference can be performed in time $O(q^2)$. To prove this, first consider the case $d(\mathbf{g}, \mathbf{g}^+, \mathbf{g}^-) = m$, where $m$ is an integer between $-q$ and $q$. In this case we can replace the loss augmented inference problem with

$$\ell'(m) + \max_{\mathbf{g},\mathbf{g}^+,\mathbf{g}^-} \left\{ \mathbf{g}^\mathsf{T} f(\mathbf{x}) + {\mathbf{g}^+}^\mathsf{T} f(\mathbf{x}^+) + {\mathbf{g}^-}^\mathsf{T} f(\mathbf{x}^-) \right\} \quad \text{s.t.} \quad d(\mathbf{g}, \mathbf{g}^+, \mathbf{g}^-) = m \,. \quad (9)$$

One can solve Eq. 9 for each possible value of $m$. It is straightforward to see that the largest of those $2q+1$ maxima is the solution to Eq. 7. Then, what remains for us is to solve Eq. 9.

To solve Eq. 9, consider the $i^{th}$ bit for each of the three codes, *i.e.* $a = \mathbf{g}[i]$, $b = \mathbf{g}^+[i]$, and $c = \mathbf{g}^-[i]$, where $\mathbf{v}[i]$ denotes the $i^{th}$ element of vector $\mathbf{v}$. There are 8 ways to select $a, b$ and $c$, but no matter what values they take on, they can only change the value of $d(\mathbf{g}, \mathbf{g}^+, \mathbf{g}^-)$ by $-1$, $0$, or $+1$. Accordingly, let $e_i \in \{-1, 0, +1\}$ denote the effect of the $i^{th}$ bits on $d(\mathbf{g}, \mathbf{g}^+, \mathbf{g}^-)$. For each value of $e_i$, we can easily compute the maximal contribution of $(a, b, c)$ to Eq. 9 by:

$$cont(i, e_i) \;=\; \max_{a,b,c} \left\{ a f(\mathbf{x})[i] + b f(\mathbf{x}^+)[i] + c f(\mathbf{x}^-)[i] \right\} \quad (10)$$

such that $a, b, c \in \{-1, +1\}$ and $\|a-b\|_H - \|a-c\|_H = e_i$.

Therefore, to solve Eq. 9, we aim to select values for $e_i$, for all $i$, such that $\sum_{i=1}^q e_i = m$ and $\sum_{i=1}^q cont(i, e_i)$ is maximized. This can be solved for any $m$ using a dynamic programming algorithm, similar to knapsack, in $O(q^2)$. Finally, we choose $m$ that maximizes Eq. 9 and set the bits to the configurations that maximized $cont(i, e_i)$.

### 3.3 Perceptron-like learning

Our learning algorithm is a form of stochastic gradient descent, where in the $t^{th}$ iteration we sample a triplet $(\mathbf{x}, \mathbf{x}^+, \mathbf{x}^-)$ from the dataset, and then take a step in the direction that decreases the upper bound on the triplet's loss in Eq. 6. To this end, we randomly initialize $\mathbf{w}^{(0)}$. Then, at each iteration $t + 1$, given $\mathbf{w}^{(t)}$, we use the following procedure to update the parameters, $\mathbf{w}^{(t+1)}$:

1. Select a random triplet $(\mathbf{x}, \mathbf{x}^+, \mathbf{x}^-)$ from dataset $\mathcal{D}$.

2. Compute $(\hat{\mathbf{h}}, \hat{\mathbf{h}}^+, \hat{\mathbf{h}}^-) = (b(\mathbf{x}; \mathbf{w}^{(t)}), b(\mathbf{x}^+; \mathbf{w}^{(t)}), b(\mathbf{x}^-; \mathbf{w}^{(t)}))$ using Eq. 5.

3. Compute $(\hat{\mathbf{g}}, \hat{\mathbf{g}}^+, \hat{\mathbf{g}}^-)$, the solution to the loss-augmented inference problem in Eq. 7 .

4. Update model parameters using

$$\mathbf{w}^{(t+1)} = \mathbf{w}^{(t)} + \eta \left[ \frac{\partial f(\mathbf{x})}{\partial \mathbf{w}} \left( \hat{\mathbf{h}} - \hat{\mathbf{g}} \right) + \frac{\partial f(\mathbf{x}^+)}{\partial \mathbf{w}} \left( \hat{\mathbf{h}}^+ - \hat{\mathbf{g}}^+ \right) + \frac{\partial f(\mathbf{x}^-)}{\partial \mathbf{w}} \left( \hat{\mathbf{h}}^- - \hat{\mathbf{g}}^- \right) - \lambda \mathbf{w}^{(t)} \right] ,$$

where $\eta$ is the learning rate, and $\partial f(\mathbf{x})/\partial \mathbf{w} \equiv \partial f(\mathbf{x}; \mathbf{w})/\partial \mathbf{w}|_{\mathbf{w}=\mathbf{w}^{(t)}} \in \mathbb{R}^{|\mathbf{w}| \times q}$ is the transpose of the Jacobian matrix, where $|\mathbf{w}|$ is the number of parameters.

This update rule can be seen as gradient descent in the upper bound of the regularized empirical loss. Although the upper bound in Eq. 6 is not differentiable at isolated points (owing to the $\max$ terms), in our experiments we find that this update rule consistently decreases both the upper bound and the actual regularized empirical loss $\mathcal{L}(\mathbf{w})$.

## 4 Asymmetric Hamming distance

When Hamming distance is used to score and retrieve the nearest neighbors to a given query, there is a high probability of a tie, where multiple items are equidistant from the query in Hamming space. To break ties and improve the similarity measure, previous work suggests the use of an *asymmetric Hamming* (AH) distance [9, 14]. With an AH distance, one stores dataset entries as binary codes (for storage efficiency) but the queries are not binarized. An asymmetric distance function is therefore defined on a real-valued query vector, $\mathbf{v} \in \mathbb{R}^q$, and a database binary code, $\mathbf{h} \in \mathcal{H}$. Computing AH distance is slightly less efficient than Hamming distance, and efficient retrieval algorithms, such as [25], are not directly applicable. Nevertheless, the AH distance can also be used to re-rank items retrieved using Hamming distance, with a negligible increase in run-time. To improve efficiency further when there are many codes to be re-ranked, AH distance from the query to binary codes can be pre-computed for each 8 or 16 consecutive bits, and stored in a query-specific lookup table.

In this work, we use the following asymmetric Hamming distance function

$$AH(\mathbf{h}, \mathbf{v}; \mathbf{s}) = \frac{1}{4} \| \mathbf{h} - \tanh(\mathrm{Diag}(\mathbf{s})\, \mathbf{v}) \|_2^2 , \tag{11}$$

where $\mathbf{s} \in \mathbb{R}^q$ is a vector of scaling parameters that control the slope of hyperbolic tangent applied to different bits; $\mathrm{Diag}(\mathbf{s})$ is a diagonal matrix with the elements of $\mathbf{s}$ on its diagonal. As the scaling factors in $\mathbf{s}$ approach infinity, AH and Hamming distances become identical. Here we use the AH distance between a database code $b(\mathbf{x}')$ and the real-valued projection for the query $f(\mathbf{x})$. Based on our validation sets, the AH distance of Eq. 11 is relatively insensitive to values in $\mathbf{s}$. For the experiments we simply use $\mathbf{s}$ to scale the average absolute values of the elements of $f(\mathbf{x})$ to be 0.25.

## 5 Implementation details

In practice, the basic learning algorithm described in Sec. 3 is implemented with several modifications. First, instead of using a single training triplet to estimate the gradients, we use mini-batches comprising 100 triplets and average the gradient. Second, for each triplet $(\mathbf{x}, \mathbf{x}^+, \mathbf{x}^-)$, we replace $\mathbf{x}^-$ with a "hard" example by selecting an item among all negative examples in the mini-batch that is closest in the current Hamming distance to $b(\mathbf{x})$. By harvesting hard negative examples, we ensure that the Hamming constraints for the triplets are not too easily satisfied. Third, to find good binary codes, we encourage each bit, averaged over the training data, to be mean-zero before quantization (motivated in [35]). This is accomplished by adding the following penalty to the objective function:

$$\frac{1}{2} \| \operatorname*{mean}_{\mathbf{x}} \big( f(\mathbf{x}; \mathbf{w}) \big) \|_2^2 , \tag{12}$$

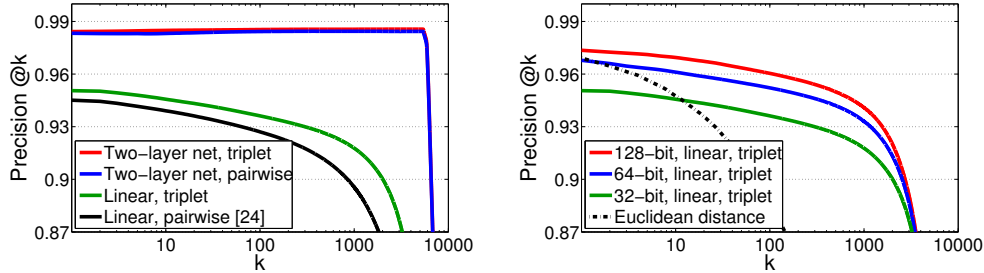

Figure 1: MNIST precision@$k$: (left) four methods (with 32-bit codes); (right) three code lengths.

where $\mathrm{mean}(f(\mathbf{x}; \mathbf{w}))$ denotes the mean of $f(\mathbf{x}; \mathbf{w})$ across the training data. In our implementation, for efficiency, the stochastic gradient of Eq. 12 is computed per mini-batch using the Jacobian matrix in the update rule (see Sec. 3.3). Empirically, we observe that including this term in the objective improves the quality of binary codes, especially with the triplet ranking loss.

We use a heuristic to adapt learning rates, known as *bold driver* [2]. For each mini-batch we evaluate the learning objective before the parameters are updated. As long as the objective is decreasing we slowly increase the learning rate $\eta$, but when the objective increases, $\eta$ is halved. In particular, after every 25 epochs, if the objective, averaged over the last 25 epochs, decreased, we increase $\eta$ by 5%, otherwise we decrease $\eta$ by 50%. We also used a momentum term; *i.e.* the previous gradient update is scaled by 0.9 and then added to the current gradient.

All experiments are run on a *GPU* for $2,000$ passes through the datasets. The training time for our current implementation is under 4 hours of GPU time for most of our experiments. The two exceptions involve CIFAR-10 with 6400-D inputs and relatively long code-lengths of 256 and 512 bits, for which the training times are approximated 8 and 16 hours respectively.

## 6   Experiments

Our experiments evaluate Hamming distance metric learning using two families of hash functions, namely, linear transforms and multilayer neural networks (see Sec. 2). For each, we examine two loss functions, the pairwise hinge loss (Eq. 2) and the triplet ranking loss (Eq. 3).

Experiments are conducted on two well-known image corpora, MNIST [1] and CIFAR-10 [19]. Ground-truth similarity labels are derived from class labels; items from the same class are deemed similar[4]. This definition of similarity ignores intra-class variations and the existence of sub-categories, *e.g.* styles of handwritten fours, or types of airplanes. Nevertheless, we use these coarse similarity labels to evaluate our framework. To that end, using items from the test set as queries, we report *precision@$k$*, *i.e.* the fraction of $k$ closest items in Hamming distance that are same-class neighbors. We also show $k$NN retrieval results for qualitative inspection. Finally, we report Hamming (H) and asymmetric Hamming (AH) $k$NN classification rates on the test sets.

**Datasets.** The MNIST [1] digit dataset contains $60,000$ training and $10,000$ test images ($28 \times 28$ pixels) of ten handwritten digits (0 to 9). Of the $60,000$ training images, we set aside $5,000$ for validation. CIFAR-10 [19] comprises $50,000$ training and $10,000$ test color images ($32 \times 32$ pixels). Each image belongs to one of 10 classes, namely airplane, automobile, bird, cat, deer, dog, frog, horse, ship, and truck. The large variability in scale, viewpoint, illumination, and background clutter poses a significant challenge for classification. Instead of using raw pixel values, we borrow a bag-of-words representation from Coates et al [6]. Its 6400-D feature vector comprises one 1600-bin histogram per image quadrant, the codewords of which are learned from $6 \times 6$ image patches. Such high-dimensional inputs are challenging for learning similarity-preserving hash functions. Of the $50,000$ training images, we set aside $5,000$ for validation.

**MNIST:** We optimize binary hash functions, mapping raw MNIST images to 32, 64, and 128-bit codes. For each test code we find the $k$ closest training codes using Hamming distance, and report precision@$k$ in Fig. 1. As one might expect, the non-linear mappings[5] significantly outperform linear mappings. We also find that the triplet loss (Eq. 3) yields better performance than the pairwise

| Hash function, Loss | Distance | $k$NN | 32 **bits** | 64 **bits** | 128 **bits** |
|---|---|---|---|---|---|
| Linear, pairwise hinge [24] | Hamming | 2 NN | 4.66 | 3.16 | 2.61 |
| Linear, triplet ranking | | 2 NN | 4.44 | 3.06 | 2.44 |
| Two-layer Net, pairwise hinge | | 30 NN | 1.50 | 1.45 | 1.44 |
| Two-layer Net, triplet ranking | | 30 NN | 1.45 | 1.38 | 1.27 |
| Linear, pairwise hinge | Asym. Hamming | 3 NN | 4.30 | 2.78 | 2.46 |
| Linear, triplet ranking | | 3 NN | 3.88 | 2.90 | 2.51 |
| Two-layer Net, pairwise hinge | | 30 NN | 1.50 | 1.36 | 1.35 |
| Two-layer Net, triplet ranking | | 30 NN | 1.45 | 1.29 | 1.20 |

| Baseline | Error |
|---|---|
| Deep neural nets with pre-training [16] | 1.2 |
| Large margin nearest neighbor [34] | 1.3 |
| RBF-kernel SVM [8] | 1.4 |
| Neural network [31] | 1.6 |
| Euclidean 3NN | 2.89 |

Table 1: Classification error rates on MNIST test set.

loss (Eq. 2). The sharp drop in precision at $k = 6000$ is a consequence of the fact that each digit in MNIST has approximately 6000 same-class neighbors. Fig. 1 (right) shows how precision improves as a function of the binary code length. Notably, $k$NN retrieval, for $k > 10$ and all code lengths, yields higher precision than Euclidean NN on the $784$-D input space. Further, note that these Euclidian results effectively provide an upper bound on the performance one would expect with existing hashing methods that preserve Eucliean distances (*e.g.*, [13, 17, 20, 35]).

One can also evaluate the fidelity of the Hamming space represenation in terms of classification performance from the Hamming codes. To focus on the quality of the hash functions, and the speed of retrieval for large-scale multimedia datasets, we use a $k$NN classifier; *i.e.* we just use the retrieved neighbors to predict class labels for each test code. Table 1 reports classification error rates using $k$NN based on Hamming and asymmetric Hamming distance. Non-linear mappings, even with only 32-bit codes, significantly outperform linear mappings (*e.g.* with 128 bits). The ranking hinge loss also improves upon the pairwise hinge loss, even though the former has no hyperparameters. Table 1 also indicates that AH distance provides a modest boost in performance. For each method the parameter $k$ in the $k$NN classifier is chosen based on the validation set.

For baseline comparison, Table 1 reports state-of-the-art performance on MNIST with sophisticated discriminative classifiers (excluding those using examplar deformations and convolutional nets). Despite the simplicity of a $k$NN classifier, our model achieves error rates of 1.29% and 1.20% using 64- and 128-bit codes. This is compared to 1.4% with RBF-SVM [8], and to 1.6%, the best published neural net result for this version of the task [31]. Our model also out performs the metric learning approach of [34], and is competitive with the best known Deep Belief Network [16]; although they used unsupervised pre-training while we do not.

The above results show that our Hamming distance metric learning framework can preserve sufficient semantic similarity, to the extent that Hamming $k$NN classification becomes competitive with state-of-the-art discriminative methods. Nevertheless, our method is not solely a classifier, and it can be used within many other machine learning algorithms.

In comparison, another hashing technique called iterative quantization (ITQ) [13] achieves 8.5% error on MNIST and 78% accuracy on CIFAR-10. Our method compares favorably, especially on MNIST. However, ITQ [13] inherently binarizes the outcome of a supervised classifier (Canonical Correlation Analysis with labels), and does not explicitly learn a similarity measure on the input features based on pairs or triplets.

**CIFAR-10:** On CIFAR-10 we optimize hash functions for 64, 128, 256, and 512-bit codes. The supplementary material includes precision@$k$ curves, showing superior quality of hash functions learned by the ranking loss compared to the pairwise loss. Here, in Fig. 2, we depict the quality of retrieval results for two queries, showing the 16 nearest neighbors using 256-bit codes, 64-bit codes (both learned with the triplet ranking loss), and Euclidean distance in the original 6400-D feature space. The number of class-based retrieval errors is much smaller in Hamming space, and the similarity in visual appearance is also superior. More such results, including failure modes, are shown in the supplementary material.

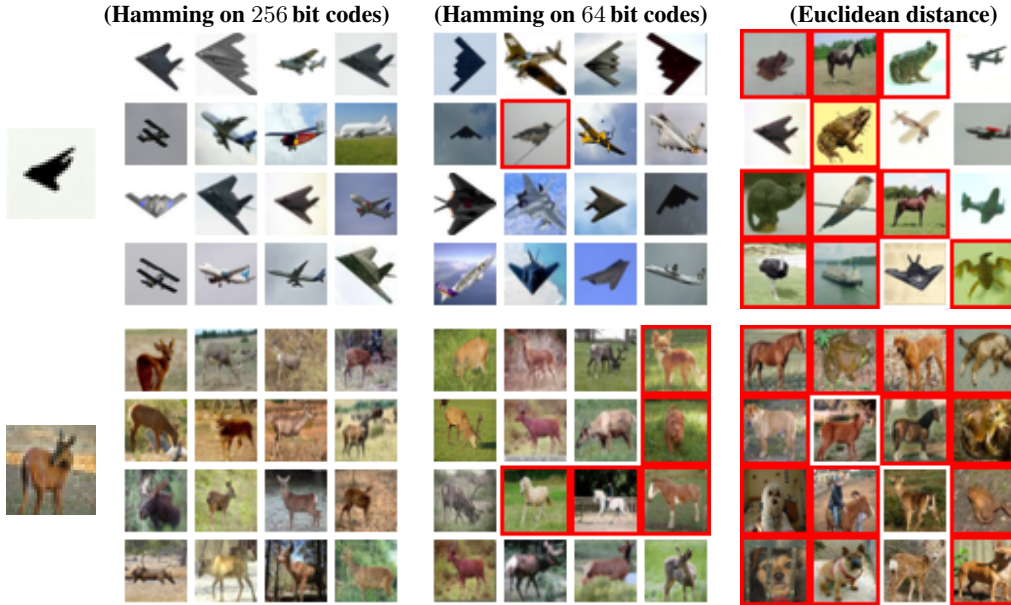

**(Hamming on** 256 **bit codes)**     **(Hamming on** 64 **bit codes)**     **(Euclidean distance)**

Figure 2: Retrieval results for two CIFAR-10 test images using Hamming distance on 256-bit and 64-bit codes, and Euclidean distance on bag-of-words features. Red rectangles indicate mistakes.

| Hashing, Loss | Distance | $k$NN | 64 **bits** | 128 **bits** | 256 **bits** | 512 **bits** |
|---|---|---|---|---|---|---|
| Linear, pairwise hinge [24] | H | 7 NN | 72.2 | 72.8 | 73.8 | 74.6 |
| Linear, pairwise hinge | AH | 8 NN | 72.3 | 73.5 | 74.3 | 74.9 |
| Linear, triplet ranking | H | 2 NN | 75.1 | 75.9 | 77.1 | 77.9 |
| Linear, triplet ranking | AH | 2 NN | 75.7 | 76.8 | 77.5 | 78.0 |

| Baseline | Accuracy |
|---|---|
| One-vs-all linear SVM [6] | 77.9 |
| Euclidean 3NN | 59.3 |

Table 2: Recognition accuracy on the CIFAR-10 test set (H $\equiv$ Hamming, AH $\equiv$ Asym. Hamming).

Table 2 reports classification performance (showing accuracy instead of error rates for consistency with previous papers). Euclidean NN on the 6400-D input features yields under 60% accuracy, while $k$NN with the binary codes obtains $76 - 78\%$. As with MNIST data, this level of performance is comparable to one-vs-all SVMs applied to the same features [6]. Not surprisingly, training fully-connected neural nets on 6400-dimensional features with only 50, 000 training examples is challenging and susceptible to over-fitting, hence the results of neural nets on CIFAR-10 were not competitive. Previous work [19] had some success training convolutional neural nets on this dataset. Note that our framework can easily incorporate convolutional neural nets, which are intuitively better suited to the intrinsic spatial structure of natural images.

## 7 Conclusion

We present a framework for Hamming distance metric learning, which entails learning a discrete mapping from the input space onto binary codes. This framework accommodates different families of hash functions, including quantized linear transforms, and multilayer neural nets. By using a piecewise-smooth upper bound on a triplet ranking loss, we optimize hash functions that are shown to preserve semantic similarity on complex datasets. In particular, our experiments show that a simple $k$NN classifier on the learned binary codes is competitive with sophisticated discriminative classifiers. While other hashing papers have used CIFAR or MNIST, none report $k$NN classification performance, often because it has been thought that the bar established by state-of-the-art classifiers is too high. On the contrary our $k$NN classification performance suggests that Hamming space can be used to represent complex semantic structures with high fidelity. One appeal of this approach is the scalability of $k$NN search on binary codes to billions of data points, and of $k$NN classification to millions of class labels.

## Footnotes

[1]For presentation clarity, in linear and nonlinear cases, we omit bias terms. They are incorporated by adding one dimension to the input vectors, and to the hidden layers of neural networks, with a fixed value of one.

[2]The Hamming norm $\|\mathbf{v}\|_H$ is defined as the number of non-zero entries of vector $\mathbf{v}$.

[3]For presentation clarity we will sometimes drop the dependence of $f$ and $b$ on $\mathbf{w}$, and write $b(\mathbf{x})$ and $f(\mathbf{x})$.

[4]Training triplets are created by taking two items from the same class, and one item from a different class.

[5]The two-layer neural nets for Fig. 1 and Table 1 had 1 hidden layer with $512$ units. Weights were initialized randomly, and the Jacobian with respect to the parameters was computed with the backprop algorithm [27].

# References

[1] http://yann.lecun.com/exdb/mnist/.

[2] R. Battiti. Accelerated backpropagation learning: Two optimization methods. *Complex Systems*, 1989.

[3] A. Bergamo, L. Torresani, and A. Fitzgibbon. Picodes: Learning a compact code for novel-category recognition. *NIPS*, 2011.

[4] M. Charikar. Similarity estimation techniques from rounding algorithms. *STOC*, 2002.

[5] G. Chechik, V. Sharma, U. Shalit, and S. Bengio. Large scale online learning of image similarity through ranking. *JMLR*, 2010.

[6] A. Coates, H. Lee, and A. Ng. An analysis of single-layer networks in unsupervised feature learning. *AISTATS*, 2011.

[7] J. Davis, B. Kulis, P. Jain, S. Sra, and I. Dhillon. Information-theoretic metric learning. *ICML*, 2007.

[8] D. Decoste and B. Schölkopf. Training invariant support vector machines. *Machine Learning*, 2002.

[9] W. Dong, M. Charikar, and K. Li. Asymmetric distance estimation with sketches for similarity search in high-dimensional spaces. *SIGIR*, 2008.

[10] P. Felzenszwalb, R. Girshick, D. McAllester, and D. Ramanan. Object detection with discriminatively trained part-based models. *IEEE Trans. PAMI*, 2010.

[11] A. Frome, Y. Singer, F. Sha, and J. Malik. Learning globally-consistent local distance functions for shape-based image retrieval and classification. *ICCV*, 2007.

[12] J. Goldberger, S. Roweis, G. Hinton, and R. Salakhutdinov. Neighbourhood components analysis. *NIPS*, 2004.

[13] Y. Gong and S. Lazebnik. Iterative quantization: A procrustean approach to learning binary codes. *CVPR*, 2011.

[14] A. Gordo and F. Perronnin. Asymmetric distances for binary embeddings. *CVPR*, 2011.

[15] D. Greene, M. Parnas, and F. Yao. Multi-index hashing for information retrieval. *FOCS*, 1994.

[16] G. Hinton and R. Salakhutdinov. Reducing the dimensionality of data with neural networks. *Science*, 2006.

[17] P. Indyk and R. Motwani. Approximate nearest neighbors: towards removing the curse of dimensionality. *STOC*, 1998.

[18] H. Jégou, M. Douze, and C. Schmid. Product quantization for nearest neighbor search. *IEEE Trans. PAMI*, 2011.

[19] A. Krizhevsky. Learning multiple layers of features from tiny images. *MSc. thesis, Univ. Toronto*, 2009.

[20] B. Kulis and T. Darrell. Learning to hash with binary reconstructive embeddings. *NIPS*, 2009.

[21] B. Kulis and K. Grauman. Kernelized locality-sensitive hashing for scalable image search. *ICCV*, 2009.

[22] W. Liu, J. Wang, R. Ji, Y. Jiang, and S. Chang. Supervised hashing with kernels. *CVPR*, 2012.

[23] M. Muja and D. Lowe. Fast approximate nearest neighbors with automatic algorithm configuration. *VISSAPP*, 2009.

[24] M. Norouzi and D. J. Fleet. Minimal Loss Hashing for Compact Binary Codes. *ICML*, 2011.

[25] M. Norouzi, A. Punjani, and D. Fleet. Fast search in hamming space with multi-index hashing. *CVPR*, 2012.

[26] M. Raginsky and S. Lazebnik. Locality-sensitive binary codes from shift-invariant kernels. *NIPS*, 2009.

[27] D. Rumelhart, G. Hinton, and R. Williams. *Learning internal representations by error propagation*. MIT Press, 1986.

[28] R. Salakhutdinov and G. Hinton. Semantic hashing. *Int. J. Approximate Reasoning*, 2009.

[29] G. Shakhnarovich, P. A. Viola, and T. Darrell. Fast pose estimation with parameter-sensitive hashing. *ICCV*, 2003.

[30] S. Shalev-Shwartz, Y. Singer, and A. Ng. Online and batch learning of pseudo-metrics. *ICML*, 2004.

[31] P. Simard, D. Steinkraus, and J. Platt. Best practice for convolutional neural networks applied to visual document analysis. *ICDR*, 2003.

[32] A. Torralba, R. Fergus, and Y. Weiss. Small codes and large image databases for recognition. *CVPR*, 2008.

[33] J. Wang, S. Kumar, and S. Chang. Sequential Projection Learning for Hashing with Compact Codes. *ICML*, 2010.

[34] K. Weinberger, J. Blitzer, and L. Saul. Distance metric learning for large margin nearest neighbor classification. *NIPS*, 2006.

[35] Y. Weiss, A. Torralba, and R. Fergus. Spectral hashing. *NIPS*, 2008.

[36] E. Xing, A. Ng, M. Jordan, and S. Russell. Distance metric learning, with application to clustering with side-information. *NIPS*, 2002.

[37] C. N. J. Yu and T. Joachims. Learning structural SVMs with latent variables. *ICML*, 2009.

